Keywords: **portfolio management, financial forecasting, recurrent neural networks**.

# Active Portfolio-Management based on Error Correction Neural Networks

**Hans Georg Zimmermann,\* Ralph Neuneier and Ralph Grothmann**
Siemens AG
Corporate Technology
D-81730 München, Germany

## Abstract

This paper deals with a neural network architecture which establishes a portfolio management system similar to the *Black / Litterman* approach. This allocation scheme distributes funds across various securities or financial markets while simultaneously complying with specific allocation constraints which meet the requirements of an investor.

The portfolio optimization algorithm is modeled by a feedforward neural network. The underlying expected return forecasts are based on error correction neural networks (ECNN), which utilize the last model error as an auxiliary input to evaluate their own misspecification.

The portfolio optimization is implemented such that (*i.*) the allocations comply with investor's constraints and that (*ii.*) the risk of the portfolio can be controlled. We demonstrate the profitability of our approach by constructing internationally diversified portfolios across 21 different financial markets of the G7 contries. It turns out, that our approach is superior to a preset benchmark portfolio.

## 1 Introduction: Portfolio-Management

We integrate the portfolio optimization algorithm suggested by Black / Litterman [1] into a neural network architecture. Combining the mean-variance theory [5] with the capital asset pricing model (CAPM) [7], this approach utilizes excess returns of the CAPM equilibrium to define a neutral, well balanced benchmark portfolio. Deviations from the benchmark allocation are only allowed within preset boundaries. Hence, as an advantage, there are no unrealistic solutions (e. g. large short positions, huge portfolio changes). Moreover, there is no need of formulating return expectations for all assets.

In contrast to Black / Litterman, excess return forecasts are estimated by time-delay recurrent error correction neural networks [8]. Investment decisions which comply with given allocation constraints are derived from these predictions. The risk exposure of the portfolio is implicitly controlled by a parameter-optimizing task over time (sec. 3 and 5).

Our approach consists of the following *three* steps: (*i.*) Construction of forecast models $f_1, \cdots, f_k$ on the basis of error correction neural networks (ECNN) for all $k$ assets (sec. 2).

(*ii.*) Computation of excess returns $e_{ij} = f_i - f_j$ by a higher-level feedforward network (sec. 3 and 4). By this, the profitability of an asset with respect to all others is measured. (*iii.*) Optimization of the investment proportions $a_i$ on the basis of the excess returns. Allocation constraints ensure, that the investment proportions $a_i$ may deviate from a given benchmark only within predefined intervals (sec. 3 and 4).

Finally, we apply our neural network based portfolio management system to an asset allocation problem concerning the *G7* countries (sec. 6).

## 2 Forecasting by Error Correction Neural Networks

Most dynamical systems are driven by a superposition of autonomous development and external influences [8]. For discrete time grids, such a dynamics can be described by a recurrent state transition $s_{t+1}$ and an output equation $y_t$ (Eq. 1).

$$
\begin{aligned}
s_{t+1} &= f(s_t, u_t, y_t - y_t^d) && \text{state transition eq.} \\
y_t &= g(s_t) && \text{output eq.}
\end{aligned}
\tag{1}
$$

The state transition $s_{t+1}$ is a mapping from the previous state $s_t$, external influences $u_t$ and a comparison between the model output $y_t$ and observed data $y_t^d$. If the last model error $(y_t - y_t^d)$ is zero, we have a perfect description of the dynamics. However, due unknown external influences $u_t$ or noise, our knowledge about the dynamics is often incomplete. Under such conditions, the model error $(y_t - y_t^d)$ quantifies the model's misfit and serves as an indicator of short-term effects or external shocks [8].

Using weight matrices $A, B, C, D$ of appropriate dimensions corresponding to $s_\tau$, $u_\tau$ and $(y_\tau - y_\tau^d)$, a neural network approach of Eq. 1 can be formulated as

$$
\begin{aligned}
s_{t+1} &= \tanh(As_t + Bu_t + D\tanh(Cs_t - y_t^d)) \\
y_t &= Cs_t
\end{aligned}
\tag{2}
$$

In Eq. 2, the output $y_t$ is recomputed by $Cs_t$ and compared to the observation $y_t^d$. Different dimensions in $s_\tau$ are adjusted by $D$. The system identification (Eq. 3) is a parameter optimization task of appropriate sized weight matrices $A, B, C, D$ [8]:

$$
\frac{1}{T} \cdot \sum_{t=1}^{T} \left(y_t - y_t^d\right)^2 \to \min_{A,B,C,D}
\tag{3}
$$

For an overview of algorithmic solution techniques see [6]. We solve the system identification task of Eq. 3 by *finite unfolding in time* using *shared weights*. For details see [3, 8]. Fig. 1 depicts the resulting neural network solution of Eq. 3.

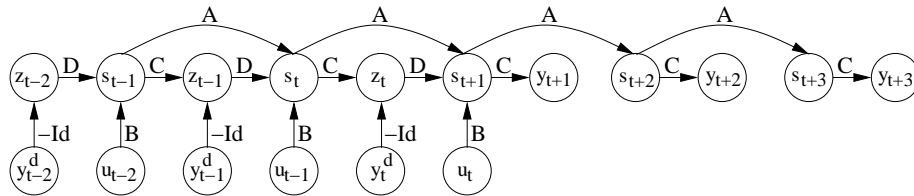

**Figure 1.** Error correction neural network (ECNN) using unfolding in time and overshooting. Note, that $-Id$ is the fixed negative of an appropriate sized identity matrix, while $z_{t-\tau}$ are output clusters with target values of zero in order to optimize the error correction mechanism.

The ECNN (Fig. 1) is best to comprehend by analyzing the dependencies of $s_t$, $u_t$, $z_t = Cs_t - y_t^d$ and $s_{t+1}$. The ECNN has two different inputs: $(i.)$ the externals $u_t$ directly influencing the state transition and $(ii.)$ the targets $y_t^d$. Only the difference between $y_t$ and $y_t^d$ has an impact on $s_{t+1}$ [8]. At all future time steps $t + \tau$, we have no compensation of the internal expectations $y_{t+\tau}$ and thus, the system offers forecasts $y_{t+\tau} = Cs_{t+\tau}$. A forecast of the ECNN is based on a modeling of the recursive structure of a dynamical system (coded in $A$), external influences (coded in $B$) and the error correction mechanism which is also acting as an external input (coded in $C$, $D$).

Using *finite* unfolding in time, we have by definition an incomplete formulation of accumulated memory in the leftmost part of the network and thus, the autoregressive modeling is handicapped [9]. Due to the error correction, the ECNN has an explicit mechanism to handle the initialization shock of the unfolding [8].

The autonomous part of the ECNN is extended into the future by the iteration of matrices $A$ and $C$. This is called *overshooting* [8]. Overshooting provides additional information about the system dynamics and regularizes the learning. Hence, the learning of false causalities might be reduced and the generalization ability of the model should be improved [8]. Of course, we have to support the additional output clusters $y_{t+\tau}$ by target values. However, due to shared weights, we have the same number of parameters [8].

## 3  The Asset Allocation Strategy

Now, we explain how the forecasts are transformed into an asset allocation vector $[a_1, \cdots, a_k]$ with investment proportions $a_i$ ($\sum a_i = 1$). For simplicity, short sales (i. e. $a_i < 0$) are *not* allowed. We have to consider, that the allocation $(i.)$ pays attention to the uncertainty of the forecasts and $(ii.)$ complies with given investment constraints.

In order to handle the uncertainty of the asset forecasts $f_i$, we utilize the concept of *excess return*. An excess return $e_{ij}$ is defined as the difference between the expected returns $f_i$ and $f_j$ of two assets $i$ and $j$, i. e. $e_{ij} = f_i - f_j$. The investment proportions $a_i$ of assets which have a superior excess return should be enlarged, because they seem to be more valuable.

Further on, let us define the cumulated excess return as a weighted sum of the excess returns for one asset $i$ over all other assets $j$,

$$e_i = \sum_{j=1}^{k} w_{ij}(f_i - f_j), \quad \text{with} \quad w_{ij} \geq 0 . \tag{4}$$

The forbiddance of short sales ($a_i \geq 0$) and the constraint, that investment proportions $a_i$ sum up to one ($\sum a_i = 1$), can be easily satisfied by the transformation

$$a_i = \frac{\exp(e_i)}{\sum_{j=1}^{k} \exp(e_j)} = a_i(w, f_1, \cdots, f_k) . \tag{5}$$

The market share constraints are given by the asset manager in form of intervals, which have a mean value of $m_i$. $m_i$ is the benchmark allocation. The admissible spread $\Delta_i$ defines how much the allocation $a_i$ may deviate from $m_i$:

$$a_i \in [m_i - \Delta_i, m_i + \Delta_i] . \tag{6}$$

Since we have to level the excess returns around the mean of the intervals, Eq. 4 is adjusted by a bias vector $v = [v_1, \cdots, v_k]$ corresponding to the benchmark allocation:

$$e_i = v_i + \sum_{j=1}^{k} w_{ij}(f_i - f_j) . \tag{7}$$

The bias $v_i$ forces the system to put funds into asset $i$, even if the cumulated excess return does not propose an investment. The vector $v$ can be computed before-hand by solving the system of nonlinear equations which results by setting the excess returns (Eq. 7) to zero:

$$
\begin{array}{rcl}
m_1 & = & a_i(v_1, \cdots, v_k) \\
\vdots & \vdots & \vdots \\
m_k & = & a_k(v_1, \cdots, v_k) \, .
\end{array}
\tag{8}
$$

Since the allocation $(m_1, \cdots, m_k)$ represents the benchmark portfolio, the pre-condition $\sum_{i=1}^{k} m_i = 1$ leads to a non-unique solution (Eq. 9) of the latter system (Eq. 8)

$$
v_i = \ln(m_i) + c,
\tag{9}
$$

for any real number $c$. In the following, we choose $c = 0$.

The interval $[m_i - \Delta_i, m_i + \Delta_i]$ defines constraints for the parameters $w_{i,1}, \ldots, w_{i,k}$ because the latter quantifies the deviation of $a_i$ from the benchmark $m_i$. Thus, the return maximization task can be stated as a constraint optimization problem with $r_{i,t}$ as the actual return of asset $i$ at time $t$:

$$
\frac{1}{T} \sum_{t=1}^{T} \sum_{i=1}^{k} r_{i,t} a_i(f_{1,t}, \cdots, f_{k,t}, w) \to \max_{w} \ \bigg| \ a_i \in [m_i - \Delta_i, m_i + \Delta_i] \, \forall i \, .
\tag{10}
$$

This problem can be solved as a penalized maximization task

$$
\frac{1}{T} \sum_{t=1}^{T} \sum_{i=1}^{k} [r_{i,t} a_i(f_{1,t}, \cdots, f_{k,t}, w) - \lambda \|a_i - m_i\|_{\Delta_i}] \to \max_{w} \, ,
\tag{11}
$$

with $\| \cdot \|_{\Delta}$ is defined as a type of $\epsilon$-insensitive error function:

$$
\|x\|_{\Delta} = \left\{ \begin{array}{cl} 0 & \text{if } |x| \le \Delta \\ |x| - \Delta & \text{otherwise} \end{array} \right.
\tag{12}
$$

Summarizing, the construction of the allocation scheme consists of the following *two* steps: (*i.*) Train the error correction sub-networks and compute the excess returns $(f_i - f_j)$. (*ii.*) Optimize the allocation parameters $w_{ij}$ using the forecast models with respect to the market share constraints $(m_1, \Delta_1, \cdots, m_k, \Delta_k)$:

$$
\frac{1}{T} \sum_{t=1}^{T} \sum_{i=1}^{k} r_{i,t} a_i(\cdots, \ln(m_i) + \sum_{j=1}^{k} w_{ij}(f_{i,t} - f_{j,t}), \cdots) \to \max_{w_{ij}} \, .
\tag{13}
$$

As we will explain in sec. 5, Eq. 13 also controls the portfolio risk.

## 4  Modeling the Asset Allocation Strategy by Neural Networks

A neural network approach of the allocation scheme (sec. 3) is shown in Fig. 2.

The first layer of the portfolio optimization neural network (Fig. 2) collects the predictions $f_i$ from the underlying ECNNs. The matrix entitled 'unfolding' computes the excess returns $e_{ij}$ for $k = 21$ assets as a contour plot. White spaces indicate weights with a value of 0, while grey equals $= 1$ and black stands for $-1$. The layer entitled 'excess returns' is designed as an output cluster, i. e. it is associated with an error function which computes error signals for each training pattern. By this, we can identify inter-market dependencies, since the neural network is forced to learn cross-market relationships.

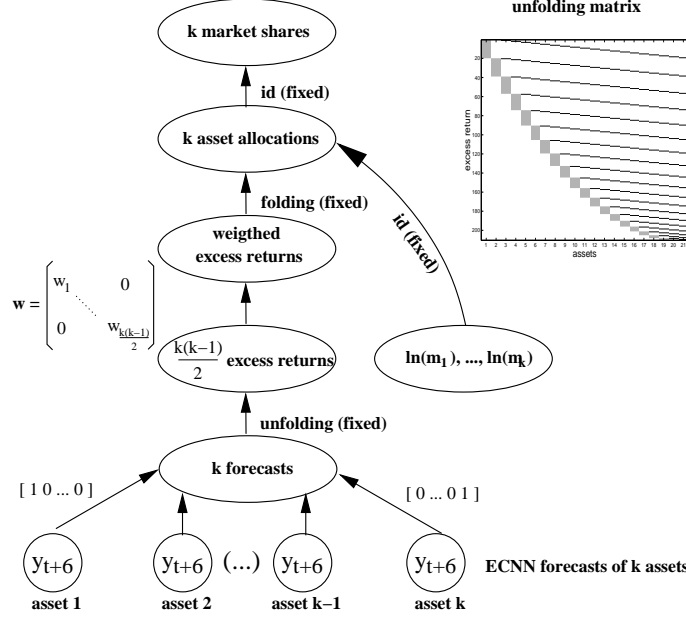

**Figure 2.** Arranged on the top of the ECNN sub-networks, a higher-level neural network models the portfolio optimization algorithm on basis of the excess returns $f_i - f_j$. The diagonal matrix which computes the weighted excess returns includes the only tunable parameters $w_{ij}$. All others are fixed.

Next, each excess return $e_{ij}$ is weighted by a particular $w_{ij}$ via a diagonal connection to the layer entitled 'weighted excess returns'. Afterwards, the weighted excess returns are folded using the transpose of the sparse matrix called 'unfolding' (see Fig. 2). By this, we calculate the sum of the weighted excess returns $\sum_j w_{ij} e_{ij}$ for each asset $i$.

According to the predictions of the excess returns $f_i - f_j$, the layer 'asset allocation' computes profitable investment decisions. In case that all excess returns $(e_1, \cdots, e_k)$ are zero the benchmark portfolio is reproduced by the offset $(v_1, \cdots, v_k) = (\ln(m_1), \cdots, \ln(m_k)))$. Otherwise, funds are allocated within the preset investment boundaries $[m_i - \Delta_i, m_i + \Delta_i]$, while simultaneously complying with the constraints $a_i \geq 0$ and $\sum a_i = 1$.

In order to prevent short selling, we assume $a_i \geq 0$ for each investment proportion. Further on, we have to guarantee that the sum of the proportions invested in the securities equals 1, i. e. $\sum a_i = 1$. Both constraints are satisfied by the activation function of the 'asset allocation' layer', which implements the non-linear transformation of Eq. 5 using *soft-max*. The return maximization task of Eq. 10 is also solved by this cluster by generating error signals utilizing the *prof-max* error function (Eq. 14).

$$\frac{1}{T} \sum_{t=1}^{T} \sum_{i=1}^{k} r_{i,t} a_i(f_{1,t}, \cdots, f_{k,t}, w) \to \max_{w} \ . \tag{14}$$

The layer 'market shares' takes care of the allocation constraints. The error function of Eq. 15 is implemented to ensure, that the investments $a_i$ do not violate preset constraints.

$$\frac{1}{T} \sum_{t=1}^{T} \sum_{i=1}^{k} [-\lambda \|a_i - m_i\|_{\Delta_i}] \to \max_{w} \ . \tag{15}$$

The 'market shares' cluster generates error signals for the penalized optimization problem stated in Eq. 11. By this, we implement a penalty for exceeding the allocation intervals $[m_i - \Delta_i, m_i + \Delta_i]$. The error signals of Eq. 10 and Eq. 11 are subsequently used for computing the gradients in order to adapt the parameters $w_{ij}$.

## 5 Risk Analysis of the Neural Portfolio-Management

In Tab. 1 we compare the mean-variance framework of Markowitz with our neural network based portfolio optimization algorithm.

|  | **Markowitz:** | **Neural Network:** |
|---|---|---|
| **input:** | for each decision: forecasts $f_i, \sigma_{ij}, \forall i, j$, accepted risk exposure $\sigma$ | prediction models $f_i(x)$, benchmark allocation $m_i$, deviation interval $\Delta_i \ \forall \ 1 \leq i \leq k$ |
| **optimization:** | $\sum_{i=1}^{k} f_i a_i \to \max_{a_i}$ with $\sum_{i,j} \sigma_{ij} a_i a_j = \sigma$ | $\sum_t \sum_i r_{i,t} a_i(f_1, \cdots, f_k, w) \to \max_w$ with implicit risk control |
| **output:** | for each decision: vector $(a_1, \cdots, a_k)$ | k decision schemes $a_i(f_1, \cdots, f_k, w)$ |

**Table 1.** Comparison of the portfolio optimization algorithm of Markowitz with our approach.

The most crucial difference between the mean-variance framework and our approach is the handling of the risk exposure (see Tab. 1). The Markowitz algorithm optimizes the expected risk explicitly by quadratic programming. Assuming that it is not possible to forecast the expected returns of the assets (often referred to as random walk hypothesis), the forecasts $f_i$ are determined by an average of most recent observed returns, while the risk-covariance matrix $[\sigma_{ij}]_{i,j=1,\cdots,k}$ is estimated by the historical volatility of the assets. Hence, the risk of the portfolio is determined by the volatility of the time series of the assets.

However, insisting on the existence of useful forecast models, we propose to derive the covariance matrix from the forecast model residuals, i. e. the risk-matrix is determined by the covariances of the model errors. Now, the risk of the portfolio is due to the non-forecastability of the assets only. Since our allocation scheme is based on the model uncertainty, we refer to this approach as *causal risk*.

Using the covariances of the model errors as a measurement of risk still allows to apply the Markowitz optimization scheme. Here, we propose to substitute the quadratic optimization problem of the Markowitz approach by the objective function of Eq. 16.

$$\frac{1}{T} \sum_{t=1}^{T} \sum_{i=1}^{k} r_{i,t} a_i \left( \ln(m_i) + \sum_{i=1}^{k} w_{ij}(f_{i,t} - f_{j,t}) \right) \to \max_w \qquad (16)$$

The error function of Eq. 16 is optimized over time $t = 1, \cdots, T$ with respect to the parameters $w_{ij}$, which are used to evaluate the certainty of the excess return forecasts. By this, it is possible to construct an asset allocations strategy which implicitly controls the risk exposure of the portfolio according to the certainty of the forecasts $f_i$. Note, that Eq. 16 can be extended by a time delay parameter $\lambda^{T-\tau}$ in order to focus on more recent events.

If the predicted excess returns $(f_{i,t} - f_{j,t})$ are reliable, then the weights $w_{ij}$ are greater than zero, because the optimization algorithm emphasizes the particular asset in comparison to

other assets with less reliable forecasts. In contrast, unreliable predictions are ruled out by pushing the associated weights $w_{ij}$ towards zero. Therefore, Eq. 16 implicitly controls the risk exposure of the portfolio although it is formulated as a return maximization task.

Eq. 16 has to be optimized with respect to the allocation constraints $m_i, \Delta_i$. This allows the definition of an *active risk* parameter $\beta \in [0, 1]$ quantifying the readiness to deviate from the benchmark portfolio $M = [m_1, \cdots, m_k]$ within the allocation constraints:

$$a_i \in [m_i - \beta \Delta_i, m_i + \beta \Delta_i] . \tag{17}$$

The weights $w_{i,j}$ and the allocations $a_i$ are now dependent on the risk level $\beta$. If $\beta = 0$, then the benchmark is recovered, while $\beta = 1$ allows deviations from the benchmark within the bounds $\Delta_i$. Thus, the active risk parameter $\beta$ analysis the risk sensitivity of the portfolios with respect to the quality of the forecast models.

## 6 Empirical Study

Now, we apply our approach to the financial markets of the *G7* countries. We work on the basis of monthly data in order to forecast the semi-annual development of the stock, cash and bond markets of the *G7* countries Spain, France, Germany, Italy, Japan, UK and USA. A separate ECNN is constructed for each market on the basis of country specific economic data. Due to the recurrent modeling, we only calculated the relative change of each input. The transformed inputs are *scaled* such that they have a mean of zero and a variance of one [8]. The complete data set (Sept. 1979 to May 1995) is divided into *three* subsets: (*i.*) Training set (Sept. 1979 to Jan. 1992). (*ii.*) Validation set (Feb. 1992 to June 1993), which is used to learn the allocation parameters $w_{ij}$. (*iii.*) Generalization set (July 1993 to May 1995). Each ECNN was trained until convergence by using stochastical *vario-eta* learning, which includes re-normalization of the gradients in each step of the backpropagation algorithm [9].

We evaluate the performance of our approach by a comparison with the benchmark portfolio $M = [m_1, \cdots, m_{21}]$ which is calculated with respect to the market shares $m_i$. The comparison of our strategy and the benchmark portfolio is drawn on the basis of the accumulated return of investment (Fig. 3). Our strategy is able to outperform the benchmark portfolio $M$ on the generalization set by nearly $10\%$. A further enhancement of the portfolio performance can only be achieved if one relaxes the market share constraints. This indicates, that the tight allocation boundaries, which prevent huge capital transactions from non-profitable to booming markets, narrow additional gains.

In Fig. 4 we compare the risk of our portfolio to the risk of the benchmark portfolio. Here, the portfolio risk is defined analogous to the mean-variance framework. However, in contrast to this approach, the expected (co-)variances are replaced by the residuals $\epsilon_i(t)$ of the underlying forecast models. The risk level which is induced by our strategy is comparable to the benchmark (Fig. 4), while simultaneously increasing the portfolio return (Fig. 3).

Fig. 5 compares the allocations of German bonds and stocks across the generalization set: A typical reciprocal investment behavior is depicted, e. g. enlarged positions in stocks often occur in parallel with smaller investments in bonds. This effect is slightly disturbed by international diversification. Not all countries show such a coherent investment behavior.

## 7 Conclusions and Future Work

We described a neural network approach which adapts the Black / Litterman portfolio optimization algorithm. Here, funds are allocated across various securities while simultaneously complying with allocation constraints. In contrast to the mean-variance theory, the risk exposure of our approach focuses on the uncertainty of the underlying forecast models.

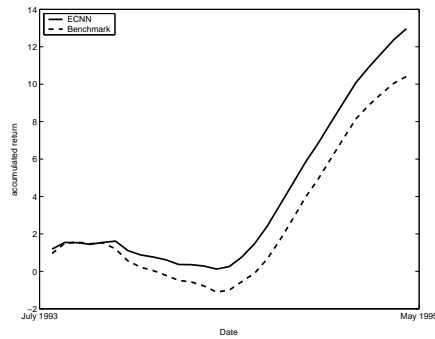

**Figure 3.**

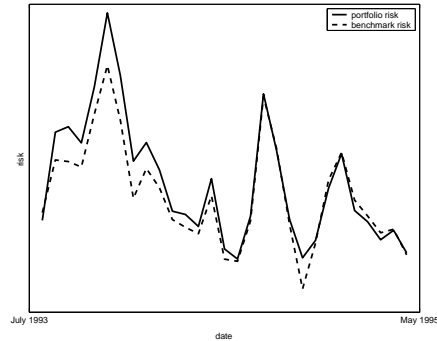

**Figure 4.**

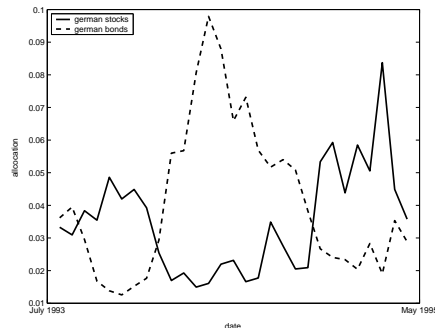

**Figure 5.**

**Fig.3.** Comparison of accumulated return of investment (generalization set).

**Fig.4.** Comparison of portfolio risk (generalization set).

**Fig.5.** Investments in German bond and stocks (generalization set).

The underlying forecasts are generated by ECNNs, since our empirical results indicate, that this is a very promising framework for financial modeling. Extending the ECNN by using techniques like overshooting, variants-invariants separation or unfolding in space and time, one is able to include additional prior knowledge of the dynamics into the model [8, 9].

Future work will include the handling of a larger universe of assets. In this case, one may extend the neural network by a bottleneck which selects the most promising assets.

## Footnotes

*To whom correspondence should be addressed: Georg.Zimmermann@mchp.siemens.de.

## References

[1] Black, F., Litterman, R.:*Global Portfolio Optimization*, Financial Analysts Journal, Sep. 1992.

[2] Elton, E. J., Gruber, M. J.: *Modern Portfolio Theory and Investment Analysis*, J. Wiley & Sons.

[3] Haykin S.: *Neural Networks. A Comprehensive Foundation.*, $2^{nd}$ ed., Macmillan, N. Y. 1998.

[4] Lintner, J.:*The Valuation of Risk Assets and the Selection of Risky Investments in Stock Portfolios and Capital Budgets*, in: Review of Economics and Statistics, Feb. 1965.

[5] Markowitz, H. M.: *Portfolio Selection*, in: Journal of Finance, Vol. 7, 1952, p. 77-91.

[6] Pearlmatter, B.:*Gradient Calculations for Dynamic Recurrent Neural Networks: A survey*, In IEEE Transactions on Neural Networks, Vol. 6, 1995.

[7] Sharpe, F.:*A Simplified Model for Portfolio Analysis*, Management Science, Vol. 9, 1963.

[8] Zimmermann, H. G., Neuneier, R., Grothmann, R.: *Modeling of Dynamical Systems by Error Correction Neural Networks*, in: Modeling and Forecasting Financial Data, Techniques of Nonlinear Dynamics, Eds. Soofi, A. and Cao, L., Kluwer 2001.

[9] Zimmermann, H.G., Neuneier, R.:*Neural Network Architectures for the Modeling of Dynamical Systems*, in: A Field Guide to Dynamical Recurrent Networks, Eds. Kremer, St. et al., IEEE.
